# Neural Computation with Winner-Take-All as the only Nonlinear Operation

**Wolfgang Maass**
Institute for Theoretical Computer Science
Technische Universität Graz
A-8010 Graz, Austria
email: maass@igi.tu-graz.ac.at
http://www.cis.tu-graz.ac.at/igi/maass

## Abstract

Everybody "knows" that neural networks need *more than a single layer of nonlinear units* to compute interesting functions. We show that *this is false* if one employs *winner-take-all* as nonlinear unit:

- Any boolean function can be computed by a *single k*-winner-take-all unit applied to weighted sums of the input variables.

- Any continuous function can be approximated arbitrarily well by a *single* soft winner-take-all unit applied to weighted sums of the input variables.

- Only positive weights are needed in these (linear) weighted sums. This may be of interest from the point of view of *neurophysiology*, since only 15% of the synapses in the cortex are inhibitory. In addition it is widely believed that there are special microcircuits in the cortex that compute winner-take-all.

- Our results support the view that winner-take-all is a very useful basic computational unit in *Neural VLSI*:
  - ☐ it is wellknown that winner-take-all of $n$ input variables can be computed very efficiently with $2n$ transistors (and a total wire length and area that is linear in $n$) in analog VLSI [Lazzaro et al., 1989]
  - ☐ we show that winner-take-all is not just useful for special purpose computations, but may serve as the only nonlinear unit for neural circuits with universal computational power
  - ☐ we show that any multi-layer perceptron needs quadratically in $n$ many gates to compute winner-take-all for $n$ input variables, hence winner-take-all provides a substantially more powerful computational unit than a perceptron (at about the same cost of implementation in analog VLSI).

Complete proofs and further details to these results can be found in [Maass, 2000].

# 1   Introduction

Computational models that involve competitive stages have so far been neglected in computational complexity theory, although they are widely used in computational brain models, artificial neural networks, and analog VLSI. The circuit of [Lazzaro et al., 1989] computes an approximate version of winner-take-all on $n$ inputs with just $2n$ transistors and wires of length $O(n)$, with lateral inhibition implemented by adding currents on a single wire of length $O(n)$. Numerous other efficient implementations of winner-take-all in analog VLSI have subsequently been produced. Among them are circuits based on silicon spiking neurons ([Meador and Hylander, 1994], [Indiveri, 1999]) and circuits that emulate attention in artificial sensory processing ([Horiuchi et al., 1997], [Indiveri,1999]). Preceding analytical results on winner-take-all circuits can be found in [Grossberg, 1973] and [Brown, 1991].

We will analyze in section 4 the computational power of the most basic competitive computational operation: winner-take-all (= 1-WTA$_n$). In section 2 we will discuss the somewhat more complex operation $k$-winner-take-all ($k$-WTA$_n$), which has also been implemented in analog VLSI [Urahama and Nagao, 1995]. Section 3 is devoted to soft winner-take-all, which has been implemented by [Indiveri, 1999] in analog VLSI via temporal coding of the output.

Our results shows that winner-take-all is a surprisingly powerful computational module in comparison with threshold gates (= McCulloch-Pitts neurons) and sigmoidal gates. Our theoretical analysis also provides answers to two basic questions that have been raised by neurophysiologists in view of the well-known asymmetry between excitatory and inhibitory connections in cortical circuits: how much computational power of neural networks is lost if only positive weights are employed in weighted linear sums, and how much learning capability is lost if only the positive weights are subject to plasticity.

# 2   Restructuring Neural Circuits with Digital Output

We investigate in this section the computational power of a $k$-winner-take-all gate computing the function      $k - WTA_n$   :   $\mathbb{R}^n \to \{0,1\}^n$

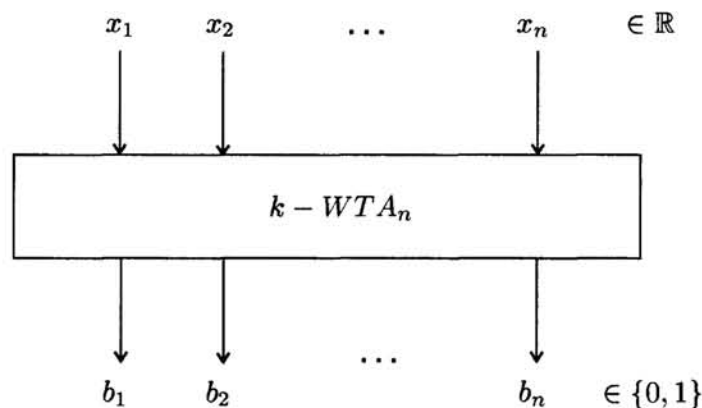

with

$b_i = 1 \leftrightarrow x_i$ is among the $k$ largest of the inputs $x_1, \dots , x_n$.

[precisely: $b_i = 1 \leftrightarrow x_j > x_i$   holds for at most $k - 1$ indices $j$]

**Theorem 1.** *Any two-layer feedforward circuit $C$ (with $m$ analog or binary input variables and one binary output variable) consisting of threshold gates (=perceptrons) can be simulated by a circuit $W$ consisting of a single k-winner-take-all gate $k\text{-}WTA_n$[1] applied to weighted sums of the input variables with positive weights. This holds for all digital inputs, and for analog inputs except for some set $S \subseteq \mathbb{R}^m$ of inputs that has measure 0.*

*In particular, any boolean function*

$$f : \{0,1\}^m \to \{0,1\}$$

*can be computed by a single k-winner-take-all gate applied to positive weighted sums of the input bits.*

**Remarks**

1. If $C$ has polynomial size and integer weights, whose size is bounded by a polynomial in $m$, then the number of linear gates $S$ in $W$ can be bounded by a polynomial in $m$, and all weights in the simulating circuit $W$ are natural numbers whose size is bounded by a polynomial in $m$.

2. The exception set of measure 0 in this result is a union of finitely many hyperplanes in $\mathbb{R}^m$. One can easily show that this exception set $S$ of measure 0 in Theorem 1 is *necessary*.

3. Any circuit that has the structure of $W$ can be converted back into a 2-layer threshold circuit, with a number of gates that is quadratic in the number of weighted sums (=linear gates) in $W$. This relies on the construction in section 4.

**Proof of Theorem 1:** Since the outputs of the gates on the hidden layer of $C$ are from $\{0,1\}$, we can assume without loss of generality that the weights $\alpha_1, \ldots, \alpha_n$ of the output gate $G$ of $C$ are from $\{-1,1\}$ (see for example [Siu et al., 1995] for details; one first observes that it suffices to use integer weights for threshold gates with binary inputs, one can then normalize these weights to values in $\{-1,1\}$ by duplicating gates on the hidden layer of $C$). Thus for any circuit input $\underline{z} \in \mathbb{R}^m$ we have $C(\underline{z}) = 1 \Leftrightarrow \sum_{j=1}^{n} \alpha_j G_j(\underline{z}) \geq \Theta$, where $G_1, \ldots, G_n$ are the threshold gates on the hidden layer of $C$, $\alpha_1, \ldots, \alpha_n$ are from $\{-1,1\}$, and $\Theta$ is the threshold of the output gate $G$. In order to eliminate the negative weights in $G$ we replace each gate $G_j$ for which $\alpha_j = -1$ by another threshold gate $\hat{G}_j$ so that $\hat{G}_j(\underline{z}) = 1 - G_j(\underline{z})$ for all $\underline{z} \in \mathbb{R}^m$ except on some hyperplane.[2] We set $\hat{G}_j := G_j$ for all $j \in \{1, \ldots, n\}$ with $\alpha_j = 1$. Then we have for all $\underline{z} \in \mathbb{R}^m$, except for $\underline{z}$ from some exception set $S$ consisting of up to $n$ hyperplanes,

$$\sum_{j=1}^{n} \alpha_j G_j(\underline{z}) = \sum_{j=1}^{n} \hat{G}_j(\underline{z}) - |\{j \in \{1, \ldots, n\} : \alpha_j = -1\}| \, .$$

Hence $C(\underline{z}) = 1 \Leftrightarrow \sum_{j=1}^{n} \hat{G}_j(\underline{z}) \geq \hat{k}$      for all $\underline{z} \in \mathbb{R}^m - S$, for some suitable $\hat{k} \in \mathbb{N}$.

Let $w_1^j, \ldots, w_m^j \in \mathbb{R}$ be the weights and $\Theta^j \in \mathbb{R}$ be the threshold of gate $\hat{G}_j, j = 1, \ldots, n$.

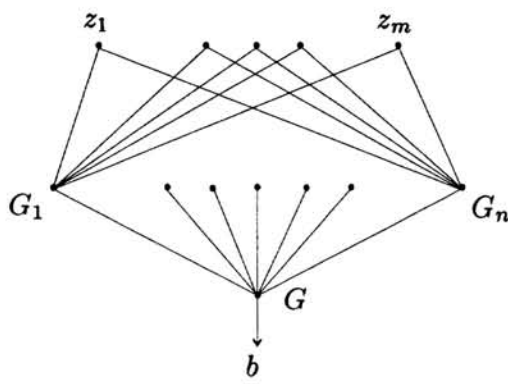

$C$

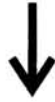

$G_1, \ldots, G_n$ are arbitrary threshold gates, $G$ is a threshold gate with weights from $\{-1,1\}$

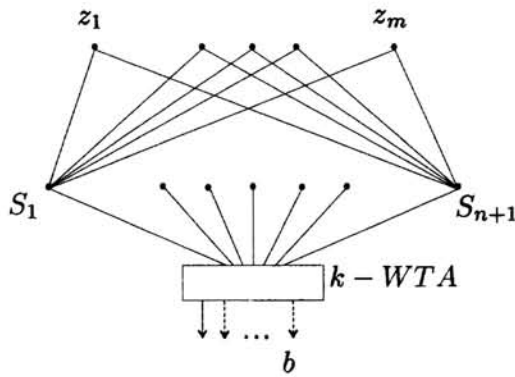

$W$

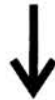

$S_1, \ldots, S_{n+1}$ are linear gates (with positive weights only, which are sums of absolute values of weights from the gates $G_1, \ldots, G_n$)

and back

Thus $\hat{G}_j(\underline{z}) = 1 \Leftrightarrow \sum\limits_{i:w_i^j > 0} |w_i^j| z_i - \sum\limits_{i:w_i^j < 0} |w_i^j| z_i \geq \Theta^j$. Hence with

$$S_j := \sum_{i:w_i^j < 0} |w_i^j| z_i + \Theta^j + \sum_{\ell \neq j} \sum_{i:w_i^\ell > 0} |w_i^\ell| z_i \quad \text{for } j = 1, \ldots, n$$

and

$$S_{n+1} := \sum_{j=1}^{n} \sum_{i:w_i^j > 0} |w_i^j| z_i$$

we have for every $j \in \{1, \ldots, n\}$ and every $\underline{z} \in \mathbb{R}^m$ :

$$S_{n+1} \geq S_j \Leftrightarrow \sum_{i:w_i^j > 0} |w_i^j| z_i - \sum_{i:w_i^j < 0} |w_i^j| z_i \geq \Theta^j \Leftrightarrow \hat{G}_j(\underline{z}) = 1 \,.$$

This implies that the $(n+1)$st output $b_{n+1}$ of the $k$-winner-take-all gate $k\text{-WTA}_{n+1}$ for

$k := n - \hat{k} + 1$ applied to $S_1, \ldots, S_{n+1}$ satisfies

$$
\begin{aligned}
b_{n+1} = 1 \quad &\Leftrightarrow |\{j \in \{1, \ldots, n+1\} : S_j > S_{n+1}\}| \leq n - \hat{k} \\
&\Leftrightarrow |\{j \in \{1, \ldots, n+1\} : S_{n+1} \geq S_j\}| \geq \hat{k} + 1 \\
&\Leftrightarrow |\{j \in \{1, \ldots, n\} : S_{n+1} \geq S_j\}| \geq \hat{k} \\
&\Leftrightarrow \sum_{j=1}^{n} \hat{G}_j(\underline{z}) \geq \hat{k} \\
&\Leftrightarrow C(\underline{z}) = 1 .
\end{aligned}
$$

Note that all the coefficients in the sums $S_1, \ldots, S_{n+1}$ are positive.  ∎

## 3 Restructuring Neural Circuits with Analog Output

In order to approximate arbitrary continuous functions with values in $[0, 1]$ by circuits that have a similar structure as those in the preceding section, we consider here a variation of a winner-take-all gate that outputs analog numbers between 0 and 1, whose values depend on the rank of the corresponding input in the linear order of all the $n$ input numbers. One may argue that such gate is no longer a "winner-take-all" gate, but in agreement with common terminology we refer to it as a *soft winner-take-all* gate. Such gate computes a function from $\mathbb{R}^n$ into $[0, 1]^n$

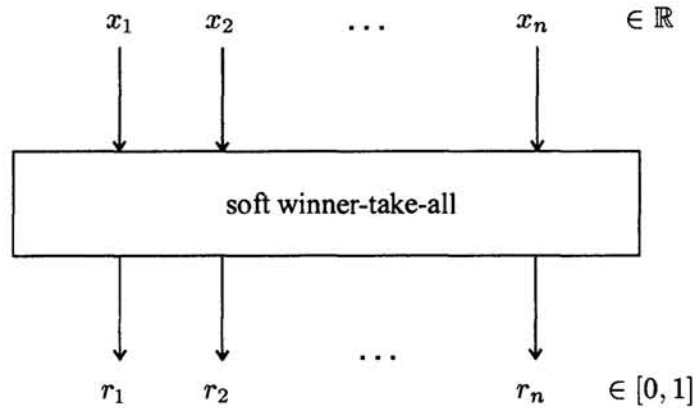

whose $i$th output $r_i \in [0, 1]$ is roughly proportional to the rank of $x_i$ among the numbers $x_1, \ldots, x_n$. More precisely: for some parameter $T \in \mathbb{N}$ we set

$$
r_i = \frac{|\{j \in \{1, \ldots, n\} : \; x_i \geq x_j\}| - \frac{n}{2}}{T} ,
$$

rounded to 0 or 1 if this value is outside $[0, 1]$. Hence this gate focuses on those inputs $x_i$ whose rank among the $n$ input numbers $x_1, \ldots, x_n$ belongs to the set $\{\frac{n}{2}, \frac{n}{2} + 1, \ldots, \min\{n, T + \frac{n}{2}\}\}$. These ranks are linearly scaled into $[0, 1]$.[3]

**Theorem 2.** *Circuits consisting of a single soft winner-take-all gate (of which we only use its first output $r_1$) applied to positive weighted sums of the input variables are universal approximators for arbitrary continuous functions from $\mathbb{R}^m$ into $[0, 1]$.*  ∎

A circuit of the type considered in Theorem 2 (with a soft winner-take-all gate applied to $n$ positive weighted sums $S_1, \ldots, S_n$) has a very simple geometrical interpretation: Over each point $\underline{z}$ of the input "plane" $\mathbb{R}^m$ we consider the relative heights of the $n$ hyperplanes $H_1, \ldots, H_n$ defined by the $n$ positive weighted sums $S_1, \ldots, S_n$. The circuit output depends only on how many of the other hyperplanes $H_2, \ldots, H_n$ are above $H_1$ at this point $\underline{z}$.

## 4  A Lower Bound Result for Winner-Take-All

One can easily see that any $k$-WTA gate with $n$ inputs can be computed by a 2-layer threshold circuit consisting of $\binom{n}{2} + n$ threshold gates:

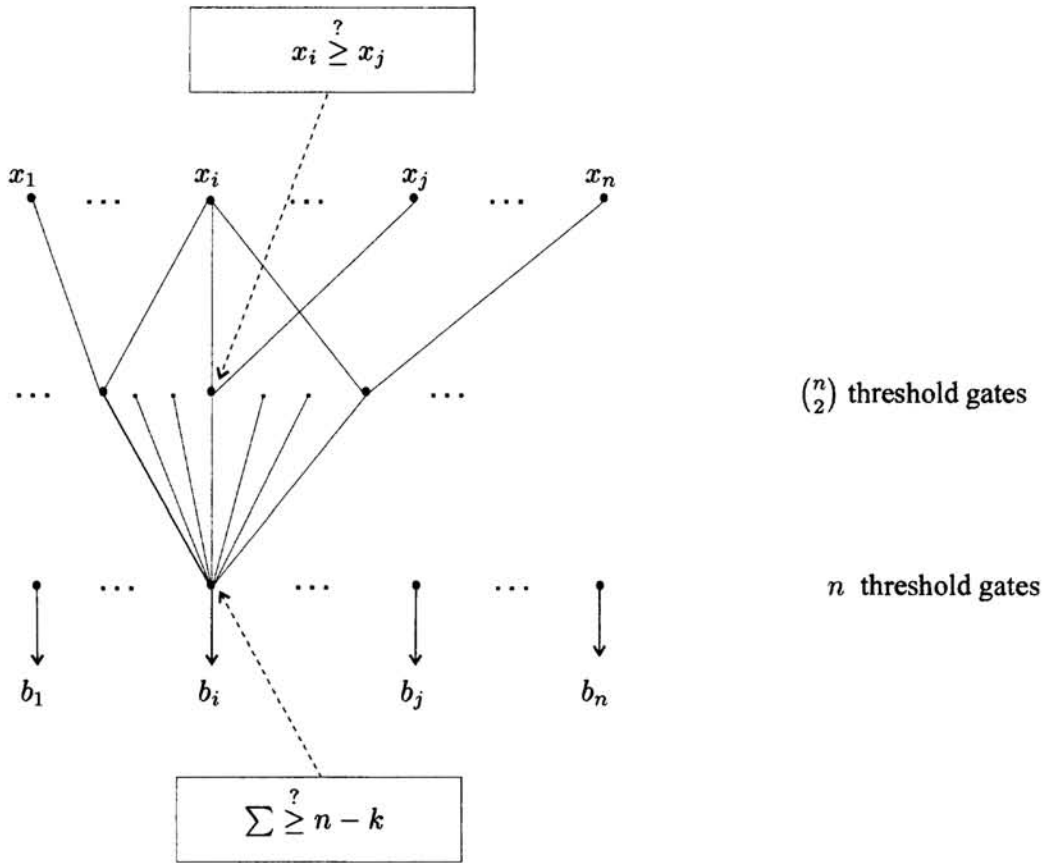

Hence the following result provides an *optimal* lower bound.

**Theorem 3.** *Any feedforward threshold circuit (=multi-layer perceptron) that computes 1-WTA for $n$ inputs needs to have at least $\binom{n}{2} + n$ gates.* ∎

## 5  Conclusions

The lower bound result of Theorem 3 shows that the computational power of winner-take-all is quite large, even if compared with the arguably most powerful gate commonly studied in circuit complexity theory: the threshold gate (also referred to a McCulloch-Pitts neuron or perceptron).

It is well known ([Minsky and Papert, 1969]) that a single threshold gate is not able to compute certain important functions, whereas circuits of moderate (i.e., polynomial) size consisting of two layers of threshold gates with polynomial size integer weights have remarkable computational power (see [Siu et al., 1995]). We have shown in Theorem 1 that any such 2-layer (i.e., 1 hidden layer) circuit can be simulated by a single $k$-winner-take-all gate, applied to polynomially many weighted sums with positive integer weights of polynomial size.

We have also analyzed the computational power of soft winner-take-all gates in the context of *analog* computation. It is shown in Theorem 2 that a single soft winner-take-all gate may serve as the only nonlinearity in a class of circuits that have universal computational power in the sense that they can approximate any continuous functions.

Furthermore our novel universal approximators require only *positive* linear operations besides soft winner-take-all, thereby showing that in principle no computational power is lost if in a biological neural system inhibition is used exclusively for unspecific lateral inhibition, and no adaptive flexibility is lost if synaptic plasticity (i.e., "learning") is restricted to excitatory synapses.

Our somewhat surprising results regarding the computational power and universality of winner-take-all point to further opportunities for low-power analog VLSI chips, since winner-take-all can be implemented very efficiently in this technology.

## Footnotes

[1] of which we only use its last output bit

[2] We exploit here that $\neg \sum_{i=1}^{m} w_i z_i \geq \Theta \Leftrightarrow \sum_{i=1}^{m} (-w_i) z_i > -\Theta$ for arbitrary $w_i, z_i, \Theta \in \mathbb{R}$ .

[3]It is shown in [Maass, 2000] that actually any continuous monotone scaling into $[0, 1]$ can be used instead.

# References

[Brown, 1991] Brown, T. X. (1991). *Neural Network Design for Switching Network Control.*. Ph.-D.-Thesis, CALTECH.

[Grossberg, 1973] Grossberg, S. (1973). Contour enhancement, short term memory, and constancies in reverberating neural networks. *Studies in Applied Mathematics*, vol. 52, 217–257.

[Horiuchi et al., 1997] Horiuchi, T. K., Morris, T. G., Koch, C., DeWeerth, S. P. (1997). Analog VLSI circuits for attention-based visual tracking. *Advances in Neural Information Processing Systems*, vol. 9, 706–712.

[Indiveri, 1999] Indiveri, G. (1999). Modeling selective attention using a neuromorphic analog VLSI device, submitted for publication.

[Lazzaro et al., 1989] Lazzaro, J., Ryckebusch, S., Mahowald, M. A., Mead, C. A. (1989). Winner-take-all networks of $O(n)$ complexity. *Advances in Neural Information Processing Systems*, vol. 1, Morgan Kaufmann (San Mateo), 703-711.

[Maass, 2000] Maass, W. (2000). On the computational power of winner-take-all, *Neural Computation*, in press.

[Meador and Hylander, 1994] Meador, J. L., and Hylander, P. D. (1994). Pulse coded winner-take-all networks. In: *Silicon Implementation of Pulse Coded Neural Networks*, Zaghloul, M. E., Meador, J., and Newcomb, R. W., eds., Kluwer Academic Publishers (Boston), 79–99.

[Minsky and Papert, 1969] Minsky, M. C., Papert, S. A. (1969). *Perceptrons*, MIT Press (Cambridge).

[Siu et al., 1995] Siu, K.-Y., Roychowdhury, V., Kailath, T. (1995). *Discrete Neural Computation: A Theoretical Foundation.* Prentice Hall (Englewood Cliffs, NJ, USA).

[Urahama and Nagao, 1995] Urahama, K., and Nagao, T. (1995). $k$-winner-take-all circuit with $O(N)$ complexity. *IEEE Trans. on Neural Networks*, vol.6, 776–778.